# Implementation Issues in the Fourier Transform Algorithm

**Yishay Mansour**[*] **Sigal Sahar**[†]
Computer Science Dept.
Tel-Aviv University
Tel-Aviv, ISRAEL

## Abstract

The Fourier transform of boolean functions has come to play an important role in proving many important learnability results. We aim to demonstrate that the Fourier transform techniques are also a useful and practical algorithm in addition to being a powerful theoretical tool. We describe the more prominent changes we have introduced to the algorithm, ones that were crucial and without which the performance of the algorithm would severely deteriorate. One of the benefits we present is the confidence level for each prediction which measures the likelihood the prediction is correct.

## 1  INTRODUCTION

Over the last few years the Fourier Transform (**FT**) representation of boolean functions has been an instrumental tool in the computational learning theory community. It has been used mainly to demonstrate the learnability of various classes of functions with respect to the uniform distribution. The first connection between the Fourier representation and learnability of boolean functions was established in [6] where the class $AC^0$ was learned (using its FT representation) in $O(n^{poly-log(n)})$ time. The work of [5] developed a very powerful algorithmic procedure: given a function and a threshold parameter it finds in polynomial time all the Fourier coefficients of the function larger than the threshold. Originally the procedure was used to learn decision trees [5], and in [8, 2, 4] it was used to learn polynomial size DNF. The FT technique applies naturally to the uniform distribution, though some of the learnability results were extended to product distribution [1, 3].

---

[*]e-mail: mansour@cs.tau.ac.il
[†]e-mail: gales@cs.tau.ac.il

A great advantage of the FT algorithm is that it does not make any assumptions on the function it is learning. We can apply it to *any* function and hope to obtain "large" Fourier coefficients. The prediction function simply computes the sum of the coefficients with the corresponding basis functions and compares the sum to some threshold. The procedure is also immune to some noise and will be able to operate even if a fraction of the examples are maliciously misclassified. Its drawback is that it requires to query the target function on randomly selected inputs.

We aim to demonstrate that the FT technique is not only a powerful theoretical tool, but also a practical one. In the process of implementing the Fourier algorithm we enhanced it in order to improve the accuracy of the hypothesis we generate while maintaining a desirable run time. We have added such feartures as the detection of inaccurate approximations "on the fly" and immediate correction of the errors incurred at a minimal cost. The methods we devised to choose the "right" parameters proved to be essential in order to achieve our goals. Furthermore, when making predictions, it is extremely beneficial to have the prediction algorithm supply an indicator that provides the confidence level we have in the prediction we made. Our algorithm provides us naturally with such an indicator as detailed in Section 4.1.

The paper is organized as follows: section 2 briefly defines the FT and describes the algorithm. In Section 3 we describe the experiments and their outcome and in Section 4 the enhancements made. We end with our conclusions in Section 5.

## 2   FOURIER TRANSFORM (FT) THEORY

In this section we briefly introduce the FT theory and algorithm. its connection to learning and the algorithm that finds the large coefficients. A comprehensive survey of the theoretical results and proofs can be found in [7].

We consider boolean functions of $n$ variables: $f : \{0,1\}^n \to \{-1,1\}$. We define the inner product: $< g, f >= 2^{-n} \sum_{x \in \{0,1\}^n} f(x)g(x) = E[g \cdot f]$, where $E$ is the expected value with respect to the uniform distribution. The basis is defined as follows: for each $z \in \{0,1\}^n$, we define the *basis function* $\chi_z(x_1, \cdots, x_n) = (-1)^{\sum_{i=1}^n x_i z_i}$. Any function of $n$ boolean inputs can be uniquely expressed as a linear combination of the basis functions. For a function $f$, the $z^{th}$ *Fourier coefficient* of $f$ is denoted by $\hat{f}(z)$, i.e., $f(x) = \sum_{z \in \{0,1\}^n} \hat{f}(z)\chi_z(x)$. The Fourier coefficients are computed by $\hat{f}(z) = < f, \chi_z >$ and we call $z$ the *coefficient-name* of $\hat{f}(z)$. We define a *t-sparse function* to be a function that has at most $t$ non-zero Fourier coefficients.

### 2.1   PREDICTION

Our aim is to approximate the target function $f$ by a $t$-sparse function $h$. In many cases $h$ will simply include the "large" coefficients of $f$. That is, if $\Lambda = \{z_1, \ldots, z_m\}$ is the set of $z$'s for which $\hat{f}(z_i)$ is "large", we set $h(x) = \sum_{z_i \in \Lambda} a_i \chi_{z_i}(x)$, where $a_i$ is our approximation of $\hat{f}(z_i)$. The hypothesis we generate using this process, $h(x)$, does not have a boolean output. In order to obtain a boolean prediction we use $Sign(h(x))$, i.e., output $+1$ if $h(x) \geq 0$ and $-1$ if $h(x) < 0$. We want to bound the error we get from approximating $f$ by $h$ using the expected error squared, $E[(f - h)^2]$. It can be shown that bounding it bounds the boolean prediction error probability, i.e., $\Pr[f(x) \neq sign(h(x))] \leq E[(f - h)^2]$. For a given $t$, the $t$-sparse

hypothesis $h$ that minimizes $E[(f-h)^2]$ simply includes the $t$ largest coefficients of $f$. Note that the more coefficients we include in our approximation and the better we approximate their values, the smaller $E[(f-h)^2]$ is going to be. This provides us with the motivation to find the "large" coefficients.

## 2.2 FINDING THE LARGE COEFFICIENTS

The algorithm that finds the "large" coefficients receives as inputs a function $f$ (a black-box it can query) and an interest threshold parameter $\theta > 0$. It outputs a list of coefficient-names that (1) includes all the coefficients-names whose corresponding coefficients are *"large"*, i.e., at least $\theta$, and (2) does not include "too many" coefficient-names. The algorithm runs in polynomial time in both $1/\theta$ and $n$.

```
SUBROUTINE search(α)
        IF TEST[f, α, θ] THEN IF |α| = n THEN OUTPUT α
                                      ELSE search(α0); search(α1);
```

Figure 1: Subroutine search

The basic idea of the algorithm is to perform a search in the space of the coefficient-names of $f$. Throughout the search algorithm (see Figure (1)) we maintain a prefix of a coefficient-name and try to estimate whether *any* of its extensions can be a coefficient-name whose value is "large". The algorithm commences by calling search($\lambda$) where $\lambda$ is the empty string. On each invocation it computes the predicate $TEST[f, \alpha, \theta]$. If the predicate is true, it recursively calls search($\alpha0$) and search($\alpha1$). Note that if $TEST$ is very permissive we may reach all the coefficients, in which case our running time will not be polynomial; its implementation is therefore of utmost interest. Formally, $TEST[f, \alpha, \theta]$ computes whether

$$E_{x\in\{0,1\}^{n-k}}E^2_{y\in\{0,1\}^k}[f(yx)\chi_\alpha(y)] \geq \theta^2, \quad \text{where } k = \|\alpha\|. \tag{1}$$

Define $f_\alpha(x) = \sum_{\beta\in\{0,1\}^{n-k}} \hat{f}(\alpha\beta)\chi_\beta(x)$. It can be shown that the expected value in (1) is exactly the sum of the squares of the coefficients whose prefix is $\alpha$, i.e., $E_{x\in\{0,1\}^{n-k}}E^2_{y\in\{0,1\}^k}[f(yx)\chi_\alpha(y)] = E_x[f_\alpha^2(x)] = \sum_{\beta\in\{0,1\}^{n-k}} \hat{f}^2(\alpha\beta)$, implying that if there exists a coefficient $|\hat{f}(\alpha\beta)| \geq \theta$, then $E[f_\alpha^2] \geq \theta^2$. This condition guarantees the correctness of our algorithm, namely that we reach all the "large" coefficients. We would like also to bound the number of recursive calls that search performs. We can show that for at most $1/\theta^2$ of the prefixes of size $k$, $TEST[f, \alpha, \theta]$ is true. This bounds the number of recursive calls in our procedure by $O(n/\theta^2)$.

In $TEST$ we would like to compute the expected value, but in order to do so efficiently we settle for an approximation of its value. This can be done as follows: (1) choose $m_1$ random $x_i \in \{0,1\}^{n-k}$, (2) choose $m_2$ random $y_{i,j} \in \{0,1\}^k$, (3) query $f$ on $y_{i,j}x_i$ (which is why we need the query model—to query $f$ on many points with the same prefix $x_i$) and receive $f(y_{i,j}x_i)$, and (4) compute the estimate as, $B_\alpha = \frac{1}{m_1}\sum_{i=1}^{m_1}\left(\frac{1}{m_2}\sum_{j=1}^{m_2} f(y_{i,j}x_i)\chi_\alpha(y_{i,j})\right)^2$. Again, for more details see [7].

## 3  EXPERIMENTS

We implemented the FT algorithm (Section 2.2) and went forth to run a series of experiments. The parameters of each experiment include the target function, $\theta$, $m_1$

and $m_2$. We briefly introduce the parameters here and defer the detailed discussion. The parameter $\theta$ determines the threshold between "small" and "large" coefficients, thus controlling the number of coefficients we will output. The parameters $\mathbf{m_1}$ and $\mathbf{m_2}$ determine how accurately we approximate the $TEST$ predicate. Failure to approximate it accurately may yield faulty, even random, results (e.g., for a ludicrous choice of $m_1 = 1$ and $m_2 = 1$) that may cause the algorithm to fail (as detailed in Section 4.3). An intelligent choice of $m_1$ and $m_2$ is therefore indispensable. This issue is discussed in greater detail in Sections 4.3 and 4.4.

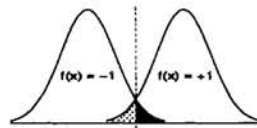

Figure 2: **Typical frequency plots and typical errors**. Errors occur in two cases: (1) the algorithm predicts a +1 response when the actual response is −1 (the lightly shaded area), and (2) the algorithm predicts a −1 response, while the true response is +1 (the darker shaded area).

Figures (3)-(5) present representative results of our experiments in the form of graphs that evaluate the output hypothesis of the algorithm on randomly chosen test points. The target function, $f$, returns a boolean response, $\pm 1$, while the FT hypothesis returns a real response. We therefore present, for each experiment, a graph constituting of two curves: the frequency of the values of the hypothesis, $h(x)$, when $f(x) = +1$, and the second curve for $f(x) = -1$. If the two curves intersect, their intersection represents the inherent error the algorithm makes.

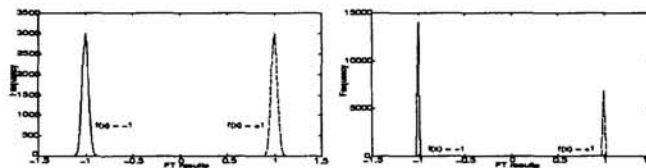

Figure 3: **Decision trees of depth 5 and 3** with 41 variables. The 5-deep (3-deep) decision tree returns −1 about 50% (62.5%) of the time. The results shown above are for values $\theta = 0.03$, $m_1 = 100$ and $m_2 = 5600$ ($\theta = 0.06$, $m_1 = 100$ and $m_2 = 1300$). Both graphs are disjoint, signifying 0% error.

# 4   RESULTS AND ALGORITHM ENHANCEMENTS

## 4.1   CONFIDENCE LEVELS

One of our most consistent and interesting empirical findings was the distribution of the error versus the value of the algorithm's hypothesis: its shape is always that of a bell shaped curve. Knowing the error distribution permits us to determine with a high (often 100%) confidence level the result for most of the instances, yielding the much sought after confidence level indicator. Though this simple logic thus far has not been supported by any theoretical result, our experimental results provide overwhelming evidence that this is indeed the case.

Let us demonstrate the strength of this technique: consider the results of the 16-term DNF portrayed in Figure (4). If the algorithm's hypothesis outputs 0.3 (translated

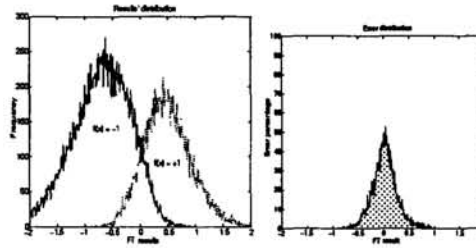

Figure 4: **16 term DNF.** This (randomly generated) DNF of 40 variables returns $-1$ about 61% of the time. The results shown above are for the values of $\theta = 0.02$, $m_2 = 12500$ and $m_1 = 100$. The hypothesis uses 186 non-zero coefficients. A total of 9.628% error was detected.

into 1 in boolean terms by the **Sign** function), we know with an 83% confidence level that the prediction is correct. If the algorithm outputs $-0.9$ as its prediction, we can virtually guarantee that the response is correct. Thus, although the total error level is over 9% we can supply a confidence level for each prediction. This is an indispensable tool for practical usage of the hypothesis.

## 4.2   DETERMINING THE THRESHOLD

Once the list of large coefficients is built and we compute the hypothesis $h(x)$, we still need to determine the threshold, $a$, to which we compare $h(x)$ (i.e., predict $+1$ iff $h(x) > a$). In the theoretical work it is assumed that $a = 0$, since a priori one cannot make a better guess. We observed that fixing $a$'s value according to our hypothesis, improves the hypothesis. $a$ is chosen to minimize the error with respect to a number of random examples.

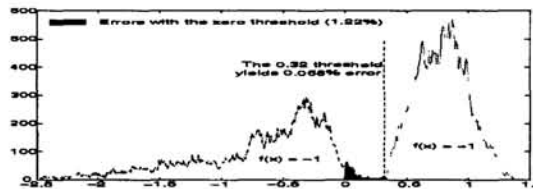

Figure 5: **8 term DNF.** This (randomly generated) DNF of 40 variables returns $-1$ about 43% of the time. The results shown above are for the values of $\theta = 0.03$, $m_2 = 5600$ and $m_1 = 100$. The hypothesis consists of 112 non-zero coefficients.

For example, when trying to learn an 8-term DNF with the zero threshold we will receive a total of 1.22% overall error as depicted in Figure (5). However, if we choose the threshold to be 0.32, we will get a diminished error of 0.068%.

## 4.3   ERROR DETECTION ON THE FLY - RETRY

During our experimentations we have noticed that at times the estimate $B_\alpha$ for $E[f_\alpha^2]$ may be inaccurate. A faulty approximation may result in the abortion of the traversal of "interesting" subtreees, thus decreasing the hypothesis' accuracy, or in traversal of "uninteresting" subtrees, thereby needlessly increasing the algorithm's runtime. Since the properties of the FT guarantee that $E[f_\alpha^2] = E[f_{\alpha 0}^2] + E[f_{\alpha 1}^2]$, we expect $B_\alpha \approx B_{\alpha 0} + B_{\alpha 1}$. Whenever this is not true, we conclude that at least one of our approximations is somewhat lacking. We can remedy the situation by

running the search procedure again on the children, i.e., *retry* node $\alpha$. This solution increases the probability of finding all the "large" coefficients. A brute force implementation may cost us an inordinate amount of time since we may retraverse subtrees that we have previously visited. However, since any discrepancies between the parent and its children are discovered—and corrected—as soon as they appear, we can circumvent any retraversal. Thus, we correct the errors without any superfluous additions to the run time.

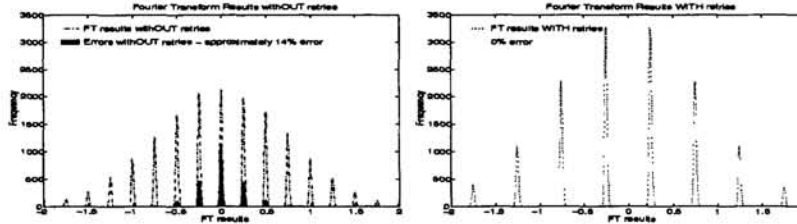

Figure 6: **Majority function** of 41 variables. The result portrayed are for values $m_1 = 100, m_2 = 800$ and $\theta = 0.08$. Note the majority-function characteristic distribution of the results[1].

We demonstrate the usefulness of this approach with an example of learning the majority function of 41 boolean variables. With*out* the retry mechanism, 8 (of a total of 42) large coefficients were missed, giving rise to 13.724% error represented by the shaded area in Figure (6). *With* the retries *all* the correct coefficients were found, yielding perfect (flawless) results represented in the dotted curve in Figure (6).

## 4.4  DETERMINING THE PARAMETERS

One of our aims was to determine the values of the different parameters, $m_1$, $m_2$ and $\theta$. Recall that in our algorithm we calculate $B_\alpha$, the approximation of $E_x[f_\alpha^2(x)]$ where $m_1$ is the number of times we sample $x$ in order to make this approximation. We sample $y$ randomly $m_2$ times to approximate $f_\alpha(x_i) = E_y[f(yx_i)\chi_\alpha(y)]$, for each $x_i$. This approximation of $f_\alpha(x_i)$ has a standard deviation of approximately $\frac{1}{\sqrt{m_2}}$. Assume that the true value is $\beta_i$, i.e. $\beta_i = f_\alpha(x_i)$, then we expect the contribution of the $i^{th}$ element to $B_\alpha$ to be $(\beta_i \pm \frac{1}{\sqrt{m_2}})^2 = \beta_i^2 \pm \frac{2\beta_i}{\sqrt{m_2}} + \frac{1}{m_2}$. The algorithm tests $B_\alpha = \frac{1}{m_1} \sum \beta_i^2 \geq \theta^2$, therefore, to ensure a low error, based on the above argument, we choose $m_2 = \frac{5}{\theta^2}$.

Choosing the right value for $m_2$ is of great importance. We have noticed on more than one occasion that *increasing* the value of $m_2$ actually *decreases* the overall run time. This is not obvious at first: seemingly, any increase in the number of times we loop in the algorithm only increases the run time. However, a more accurate value for $m_2$ means a more accurate approximation of the $TEST$ predicate, and therefore less chance of redundant recursive calls (the run time is linear in the number of recursive calls). We can see this exemplified in Figure (7) where the number of recursive calls increase drastically as $m_2$ *decreases*. In order to present Figure (7),

we learned the same 3 term DNF always using $\theta = 0.05$ and $m_1 * m_2 = 100000$. The trials differ in the specific values chosen in each trial for $m_2$.

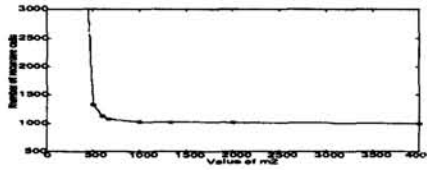

Figure 7: **Determining $m_2$**. Note that the number of recursive calls grows dramatically as $m_2$'s value decreases. For example, for $m_2 = 400$, the number of recursive calls is $14,433$ compared with only $1,329$ recursive calls for $m_2 = 500$.

**SPECIAL CASES:** When $k = \|\alpha\|$ is either very small or very large, the values we choose for $m_1$ and $m_2$ can be self-defeating: when $k \sim n$ we still loop $m_1$ ($\gg 2^{n-k}$) times, though often without gaining additional information. The same holds for very small values of $k$, and the corresponding $m_2$ ($\gg 2^k$) values. We therefore add the following feature: for small and large values of $k$ we calculate exactly the expected value thereby decreasing the run time and increasing accuracy.

# 5   CONCLUSIONS

In this work we implemented the FT algorithm and showed it to be a useful practical tool as well as a powerful theoretical technique. We reviewed major enhancements the algorithm underwent during the process. The algorithm successfully recovers functions in a reasonable amount of time. Furthermore, we have shown that the algorithm naturally derives a confidence parameter. This parameter enables the user in many cases to conclude that the prediction received is accurate with extremely high probability, even if the overall error probability is not negligible.

## Acknowledgements

This research was supported in part by The Israel Science Foundation administered by The Israel Academy of Science and Humanities and by a grant of the Israeli Ministry of Science and Technology.

## Footnotes

[1] The "peaked" distribution of the results is not coincidental. The FT of the majority function has 42 large *equal* coefficients, labeled $c_{maj}$: one for each singleton (a vector of the form 0..010..0) and one for parity (the all-ones vector). The zeros of an input vector with $z$ zeros we will contribute $\pm|(2z - 41) * c_{maj}|$ to the result and the parity will contribute $\pm c_{maj}$ (depending on whether $z$ is odd or even), so that the total contribution is an even factor of $c_{maj}$. Since $c_{maj} = \binom{40}{20}\frac{1}{2^{40}} \sim 0.12$, we have peaks around factors of 0.24. The distribution around the peaks is due to the fact we only approximate each coefficient and get a value close to $c_{maj}$.

## References

[1] Mihir Bellare. A technique for upper bounding the spectral norm with applications to learning. In $5^{th}$ *Annual Workshop on Computational Learning Theory*, pages 62–70, July 1992.

[2] Avrim Blum, Merrick Furst, Jeffrey Jackson, Michael Kearns, Yishay Mansour, and Steven Rudich. Weakly learning DNF and characterizing statistical query learning using fourier analysis. In *The $26^{th}$ Annual ACM Symposium on Theory of Computing*, pages 253 – 262, 1994.

[3] Merrick L. Furst, Jeffrey C. Jackson, and Sean W. Smith. Improved learning of $AC^0$ functions. In $4^{th}$ *Annual Workshop on Computational Learning Theory*, pages 317–325, August 1991.

[4] J. Jackson. An efficient membership-query algorithm for learning DNF with respect to the uniform distribution. In *Annual Symposium on Switching and Automata Theory*, pages 42 – 53, 1994.

[5] E. Kushilevitz and Y. Mansour. Learning decision trees using the fourier spectrum. *SIAM Journal on Computing* 22(6): 1331–1348, 1993.

[6] N. Linial, Y. Mansour, and N. Nisan. Constant depth circuits, fourier transform and learnability. *JACM* 40(3):607–620, 1993.

[7] Y. Mansour. Learning Boolean Functions via the Fourier Transform. *Advances in Neural Computation*, edited by V.P. Roychodhury and K-Y. Siu and A. Orlitsky, Kluwer Academic Pub. 1994. Can be accessed via ftp://ftp.math.tau.ac.il/pub/mansour/PAPERS/LEARNING/fourier-survey.ps.Z.

[8] Yishay Mansour. An $o(n^{\log \log n})$ learning algorihm for DNF under the uniform distribution. *J. of Computer and System Science*, 50(3):543–550, 1995.